# Some Estimates of Necessary Number of Connections and Hidden Units for Feed-Forward Networks

**Adam Kowalczyk**
Telecom Australia, Research Laboratories
770 Blackburn Road, Clayton, Vic. 3168, Australia
(a.kowalczyk@trl.oz.au)

## Abstract

The feed-forward networks with fixed hidden units (*FHU-networks*) are compared against the category of remaining feed-forward networks with variable hidden units (*VHU-networks*). Two broad classes of tasks on a finite domain $\mathbf{X} \subset \mathbf{R}^n$ are considered: approximation of every function from an open subset of functions on $\mathbf{X}$ and representation of every dichotomy of $\mathbf{X}$. For the first task it is found that both network categories require the same minimal number of synaptic weights. For the second task and $\mathbf{X}$ in general position it is shown that VHU-networks with threshold logic hidden units can have approximately $1/n$ times fewer hidden units than any FHU-network must have.

## 1   Introduction

A good candidate artificial neural network for short term memory needs to be: (*i*) easy to train, (*ii*) able to support a broad range of tasks in a domain of interest and (*iii*) simple to implement. The class of feed-forward networks with fixed hidden units (*HU*) and adjustable synaptic weights at the top layer only (shortly: *FHU-networks*) is an obvious candidate to consider in this context. This class covers a wide range of networks considered in the past, including the classical perceptron, higher order networks and non-linear associative mapping. Also a number of training algorithms were specifically devoted to this category (e.g. perceptron, madaline

or pseudoinverse) and a number of hardware solutions were investigated for their implementation (e.g. optical devices [8]).

Leaving aside the non-trivial tasks of constructing the domain specific HU for a FHU-network [9] and then optimal loading of specific tasks, in this paper we concentrate on assessing the abilities of such structures to support a wide range of tasks in comparison to more complex feedforward networks with multiple layers of variable HU (*VHU-networks*). More precisely, on a finite domain $\mathbf{X}$ two benchmark tests are considered: approximation of every function from an open subset of functions on $\mathbf{X}$ and representation of every dichotomy of $\mathbf{X}$. Some necessary and sufficient estimates of minimal necessary numbers of adaptable synaptic weights and of HU are obtained and then combined with some sufficient estimates in [10] to provide the final results. In Appendix we present an outline some of our recent results on the extension of the classical Function-Counting Theorem [2] to the multilayer case and discuss some of its implications to assessing network capacities.

## 2    Statement of the main results

In this paper $\mathbf{X}$ will denote a subset of $\mathbf{R}^n$ of $N$ points. Of interest to us are multilayer feed-forward networks (shortly *FF-networks*), $F_{\mathbf{w}} : \mathbf{X} \to \mathbf{R}$, depending on the $k$-tuple $\mathbf{w} = (w_1, ..., w_k) \in \mathbf{R}^k$ of *adjustable synaptic weights* to be selected on loading to the network desired tasks. The FF-networks are split into two categories defined above:

- FHU-network with fixed hidden units $\phi_i : \mathbf{X} \to \mathbf{R}$

$$F_{\mathbf{w}}(\mathbf{x}) \stackrel{\text{def}}{=} \sum_{i=1}^{k} w_i \phi_i(\mathbf{x}) \quad (\mathbf{x} \in \mathbf{X}), \tag{1}$$

- VHU-networks with variable hidden units $\psi_{\mathbf{w}'',i} : \mathbf{X} \to \mathbf{R}$ depending on some adjustable synaptic weights $\mathbf{w}''$, where $\mathbf{w} = (\mathbf{w}', \mathbf{w}'') \in \mathbf{R}^{k'} \times \mathbf{R}^{k''} = \mathbf{R}^k$

$$F_{\mathbf{w}}(\mathbf{x}) \stackrel{\text{def}}{=} \sum_{i=1}^{k'} w_i' \psi_{\mathbf{w}'',i}(\mathbf{x}) \quad (\mathbf{x} \in \mathbf{X}). \tag{2}$$

Of special interest are situations where hidden units are built from one or more layers of *artificial neurons*, which, for simplicity, can be thought of as devices computing simple functions of the form

$$(y_1, ..., y_m) \in \mathbf{R}^m \mapsto \sigma(w_{i_1} y_1 + w_{i_2} y_2 + \cdots + w_{i_m} y_m),$$

where $\sigma : \mathbf{R} \to \mathbf{R}$ is a non-decreasing *squashing function*. Two particular examples of squashing functions are (*i*) infinitely differentiable *sigmoid function* $t \mapsto (1 + \exp(-t))^{-1}$ and (*ii*) the *step function* $\theta(t)$ defined as 1 for $t \geq 0$ and $= 0$, otherwise. In the latter case the artificial neuron is called a *threshold logic neuron (ThL-neuron)*.

In the formulation of results below all biases are treated as synaptic weights attached to links from special constant HUs ($\equiv 1$).

## 2.1  Function approximation

The space $\mathbf{R}^{\mathbf{X}}$ of all real functions on $\mathbf{X}$ has the natural structure of a vector space isomorphic with $\mathbf{R}^N$. We introduce the euclidean norm $\|f\| \overset{\text{def}}{=} (\sum_{x \in \mathbf{X}} f^2(x))^{1/2}$ on $\mathbf{R}^{\mathbf{X}}$ and denote by $\mathcal{U} \subset \mathbf{R}^{\mathbf{X}}$ an open, non-empty subset. We say that the FF-network $F_{\mathbf{w}}$ can approximate a function $f$ on $\mathbf{X}$ with accuracy $\epsilon > 0$ if $\|f - F_{\mathbf{w}}\| < \epsilon$ for a weight vector $\mathbf{w} \in \mathbf{R}^k$.

**Theorem 1** *Assume the FF-network $F_{\mathbf{w}}$ is continuously differentiable with respect to the adjustable synaptic weights $\mathbf{w} \in \mathbf{R}^k$ and $k < N$. If it can approximate any function in $\mathcal{U}$ with any accuracy then for almost every function $f \in \mathcal{U}$, if $\lim_{i \to \infty} \|F_{\mathbf{w}(i)} - f\| = 0$, where $\mathbf{w}(1), \mathbf{w}(2), ... \in \mathbf{R}^k$, then $\lim_{i \to \infty} \|\mathbf{w}(i)\| = \infty$.*

In the above theorem "almost every" means with the exception of a subset of the Lebesgue measure 0 on $\mathbf{R}^{\mathbf{X}} \approx \mathbf{R}^N$. The proof of this theorem relies on use of Sard's theorem from differential topology (c.f. Section 3). Note that the above theorem is applicable in particular to the popular "back-propagation" network which is typically built from artificial neurons with the continuously differentiable sigmoid squashing function.

The proof of the following theorem uses a different approach, since the network is not differentiably dependent on its synaptic weights to HUs. This theorem applies in particular to the classical FF-networks built from ThL-neurons.

**Theorem 2** *A FF-network $F_{\mathbf{w}}$ must have $\geq N$ HU in the top hidden layer if all units of this layer have a finite number of activation levels and the network can approximate any function in $\mathcal{U}$ with any accuracy.*

The above theorems mean in particular that if we want to achieve an arbitrarily good approximation of any function in $\mathcal{U} \overset{\text{def}}{=} \{f : \mathbf{X} \to \mathbf{R} ; |f(\mathbf{x})| < A\}$, where $A > 0$, and we can use one of VHU-networks of the above type with synaptic weights of a *restricted magnitude only*, then we have to have at least $N$ such weights. However that many weights are necessary and sufficient to achieve the same, with a FHU-network (1) if the functions $\phi_i$ are linearly independent on $\mathbf{X}$. So variable hidden units give no advantage in this case.

## 2.2  Implementation of dichotomy

We say that the FF-network $F_{\mathbf{w}}$ can implement a dichotomy $(\mathbf{X}_-, \mathbf{X}_+)$ of $\mathbf{X}$ if there exists $\mathbf{w} \in \mathbf{R}^k$ such that $F_{\mathbf{w}} < 0$ on $\mathbf{X}_-$ and $F_{\mathbf{w}} > 0$ on $\mathbf{X}_+$.

**Proposition 3** *A FHU-network $F_{\mathbf{w}}$ can implement every dichotomy of $\mathbf{X}$ if and only if it can exactly compute every function on $\mathbf{X}$. In such a case it must have $\geq N$ HU in the top hidden layer.*

The non-trivial part of the above theorem is necessity in the first part of it, i.e. that being able to implement every dichotomy on $\mathbf{X}$ requires $N$ (fixed) hidden units. In Section 3.3 we obtain this proposition from a stronger result. Note that the above

proposition can be deduced from the classical Function-Counting Theorem [2] and also that an equivalent result is proved directly in [3, Theorem 7.2].

We say that the points of a subdomain $\mathbf{X} \subset \mathbf{R}^n$ are in *general position* if every in $\mathbf{R}^n$ contains no more than $n$ points of $\mathbf{X}$. Note that points of every finite subdomain of $\mathbf{R}^n$ are in general position after a sufficiently small perturbation and that the property of being in general position is preserved under sufficiently small perturbations. Note also that the points of a *typical* $N$-point subdomain $\mathbf{X} \subset \mathbf{R}^n$ are in general position, where "typical" means with the exception of subdomains $\mathbf{X}$ corresponding to a certain subset of Lebesgue measure 0 in the space $(\mathbf{R}^n)^N$ of all $N$-tuples of points from $\mathbf{R}^n$.

It is proved in [10] that for a subdomain set $\mathbf{X} \subset \mathbf{R}^n$ of $N$ points in general position a VHU-network having $\lceil (N-1)/n \rceil$ (adjustable) ThL-neurons in the first (and the only) hidden layer can implement every dichotomy of $\mathbf{X}$, where the notation $\lceil t \rceil$ denotes the smallest integer $\geq t$. Furthermore, examples are given showing that the above bound is tight. (Note that this paper corrects and gives rigorous proofs of some early results in [1, Lemma 1 and Theorem 1] and also improves [6, Theorem 4].) Combining these results with Proposition 3 we get the following result.

**Theorem 4** *Assume that all $N$ points of $\mathbf{X} \subset \mathbf{R}^n$ are in general position. In the class of all FF-networks which can implement every dichotomy on $\mathbf{X}$ there exists a VHU-network with threshold logic HU having a fraction $1/n + O(1/N)$ of the number of the HU that any FHU-network in this class must have. There are examples of $\mathbf{X}$ in general position of any even cardinality $N > 0$ showing that this estimate is tight.*

## 3  Proofs

Below we identify functions $f : \mathbf{X} \to \mathbf{R}$ with $N$-tuples of their values at $N$-points of $\mathbf{X}$ (ordered in a unique manner). Under this identification the FF-networks $F_\mathbf{w}$ can be regarded as a transformation

$$\mathbf{w} \in \mathbf{R}^k \to F_\mathbf{w} \in \mathbf{R}^N \tag{3}$$

with the *range* $\mathcal{R}(F_\mathbf{w}) \overset{\text{def}}{=} \{F_\mathbf{w} \; ; \; \mathbf{w} \in \mathbf{R}^k\} \subset \mathbf{R}^N$.

### 3.1  Proof of Theorem 1.

In this case the transformation (3) is continuously differentiable. Every value of it is singular since $k < N$, thus according to Sard's Theorem [5], $\mathcal{R}(F_\mathbf{w}) \subset \mathbf{R}^N$ has Lebesgue measure 0. It is enough to show now that if

$$f \in \mathcal{U} - \mathcal{R}(F_\mathbf{w}) \tag{4}$$

and

$$\lim_{i \to \infty} \|F_{\mathbf{w}(i)} - f\| = 0 \quad \text{and} \quad \|\mathbf{w}(i)\| < M, \tag{5}$$

for some $M > 0$, then a contradiction follows. Actually from (5) it follows that $f$ belongs to the topological closure $cl(\mathcal{R}_M)$ of $\mathcal{R}_M \overset{\text{def}}{=} \{F_\mathbf{w} \; ; \; \mathbf{w} \in \mathbf{R}^k \; \& \; \|\mathbf{w}\| \leq$

$M$}. However, $\mathcal{R}_M$ is a compact set as a continuous image of a closed ball {$\mathbf{w} \in \mathbf{R}^k$ ; $||\mathbf{w}|| \leq M$}, so $cl(\mathcal{R}_M) = \mathcal{R}_M$. Consequently $f \in \mathcal{R}_M \subset \mathcal{R}(F_\mathbf{w})$ which contradicts (4). Q.E.D.

## 3.2    Proof of Theorem 2.

We consider the FF-network (1) for which there exists a finite set $V \subset \mathbf{R}$ of $s$ points such that $\psi_{\mathbf{w}'',i}(\mathbf{x}) \in V$ for every $\mathbf{w}'' \in \mathbf{R}^{k''}$, $1 \leq i \leq k'$ and $\mathbf{x} \in \mathbf{X}$. It is sufficient to show that the set $\mathcal{R}(F_\mathbf{w})$ of all functions computable by $F_\mathbf{w}$ is not dense in $\mathcal{U}$ if $k' < N$ . Actually, we can write $\mathcal{R}(F_\mathbf{w})$ as a union

$$\mathcal{R}(F_\mathbf{w}) = \bigcup_{\mathbf{w}'' \in \mathbf{R}^{k''}} L_{\mathbf{w}''} \subset \mathbf{R}^N, \tag{6}$$

where each $L_{\mathbf{w}''} \overset{\text{def}}{=} \{\sum_{i=1}^{k'} w'_i \psi_{\mathbf{w}'',i} \ ; \ w'_1, ..., w'_{k'} \in \mathbf{R}\} \subset \mathbf{R}^N$ is a linear subspace of dimension $\leq k' \leq N$ uniquely determined by the vectors $\psi_{\mathbf{w}'',i} \in V^N \subset \mathbf{R}^N$, $i = 1, ..., k'$. However there is a finite number ($\leq s^N$) of different vectors in $V^N$, thus there is only a finite number ($\leq s^{Nk}$) of different linear subspaces in the family $\{L_{\mathbf{w}''} \ ; \ \mathbf{w}'' \in \mathbf{R}^{k''}\}$. Hence, as $k' < N$, the union (6) is a closed no-where dense subset of $\mathbf{R}^N$ as a finite union of proper linear subspaces (each of which is a closed and nowhere dense subset). Q.E.D.

## 3.3    Proof of Proposition 3.

We state first a stronger result. We say that a set $L$ of functions on $\mathbf{X}$ is convex if for any couple of functions $\phi_1, \phi_2$ on $\mathbf{X}$ any $\alpha > 0$, $\beta > 0$, $\alpha + \beta = 1$, the function $\alpha\phi_1 + \beta\phi_2$ also belongs to $L$.

**Proposition 5** *Let $L$ be a convex set of functions on $\mathbf{X} = \{\mathbf{x}_1, \mathbf{x}_2, ..., \mathbf{x}_N\}$ implementing every dichotomy of $\mathbf{X}$. Then for each $i \in \{1, 2, ..., N\}$ there exists a function $\phi^i \in L$ such that $\phi^i(\mathbf{x}_i) \neq 0$ and $\phi^i(\mathbf{x}_j) = 0$ for $1 \leq i \neq j \leq N$.*

**Proof.** We define a transformation $\text{SGN} : \mathbf{R}^\mathbf{X} \rightarrow \{-1, 0, +1\}^N$

$$\text{SGN}(\phi) \overset{\text{def}}{=} (\text{sgn}(\phi(\mathbf{x}_1)), \text{sgn}(\phi(\mathbf{x}_2)), ..., \text{sgn}(\phi(\mathbf{x}_N))) \in \{-1, 0, +1\}^N,$$

where $\text{sgn}(\xi) \overset{\text{def}}{=} -1$ if $\xi < 0$, $\text{sgn}(0) \overset{\text{def}}{=} 0$ and $\text{sgn}(\xi) \overset{\text{def}}{=} +1$ if $\xi > 0$. We denote by $W_k$ the subset of $\{-1, 0, +1\}^N$ of all points $\mathbf{q} = (q_1, ..., q_N)$ such that $\sum_{i=1}^N |q_i| = k$, for $k = 0, 1, ..., N$.

We show first that convexity of $L$ implies for $k \in \{1, 2, ..., N\}$ the following

$$W_k \subset \text{SGN}(L) \quad \Rightarrow \quad W_{k-1} \subset \text{SGN}(L). \tag{7}$$

For the proof assume $W_k \subset \text{SGN}(L)$ and $\mathbf{q} = (q_1, ..., q_N) \in \{-1, 0, +1\}^N$ is such that $\sum_{i=1}^N |q_i| = k - 1$. We need to show that there exists $\phi \in L$ such that

$$\text{SGN}(\phi) = \mathbf{q}. \tag{8}$$

The vector $\mathbf{q}$ has at least one vanishing entry, say, without loss of generality, $q_1 = 0$. Let $\phi^+$ and $\phi^-$ be two functions in $L$ such that

$$\mathrm{SGN}(\phi^+) = \mathbf{q}^+ \stackrel{\text{def}}{=} (+1, q_2, ..., q_N),$$
$$\mathrm{SGN}(\phi^-) = \mathbf{q}^- \stackrel{\text{def}}{=} (-1, q_2, ..., q_N).$$

Such $\phi^+$ and $\phi^-$ exist since $\mathbf{q}^+, \mathbf{q}^- \in W_k$. The function

$$\phi \stackrel{\text{def}}{=} (|\phi^+(\mathbf{x}_1)|\phi^- + |\phi^-(\mathbf{x}_1)|\phi^+)/(|\phi^+(\mathbf{x}_1)| + |\phi^-(\mathbf{x}_1)|)$$

belongs to $\mathbf{L}$ as a convex combination of two functions from $\mathbf{L}$ and satisfies (8).

Now note that the assumptions of the proposition imply that $W_N \subset \mathrm{SGN}(\mathbf{L})$. Applying (7) repeatedly we find that $W_1 \subset \mathrm{SGN}(\mathbf{L})$, which means that for every index $i$, $1 \le i \le N$, there exists a function $\phi^i \in \mathbf{L}$ with vanishing all entries but the $i$-th one. Q.E.D.

Now let us see how Proposition 3 follows from the above result. Sufficiency is obvious. For the necessity we observe that the family $F_{\mathbf{W}}$ of functions on $\mathbf{X}$ is convex being a linear space in the case of a FHU-network (1). Now if this network can compute every dichotomy of $\mathbf{X}$, then each function $\phi^i$ as in Proposition 5 equals to $F_{\mathbf{w}_i}$ for some $\mathbf{w}_i \in \mathbf{R}^k$. Thus $\mathcal{R}(F_{\mathbf{W}}) = \mathbf{R}^N$ since those functions make a basis of $\mathbf{R}^{\mathbf{X}} \approx \mathbf{R}^N$. Q.E.D.

## 4    Discussion of results

Theorem 1 combined with observations in [4] allows us to make the following contribution to the recent controversy on relevance/irrelevance of Kolmogorov's theorem on representation of continuous functions $I^n \to \mathbf{R}$, $I \stackrel{\text{def}}{=} [0, 1]$ (c.f. [4, 7]), since $I^n$ contains subsets of any cardinality.

> *The FF-networks for approximations of continuous functions on $I^n$ of rising accuracy have to be complex, at least in one of the following ways:*
>
> - *involve adjustment of a diverging number of synaptic weights and hidden units, or*
> - *require adjustment of synaptic weights of diverging magnitude, or*
> - *involve selection of "pathological" squashing functions.*

Thus one can only shift complexity from one kind to another, but not eliminate it completely. Although on theoretical grounds one can easily argue the virtues and simplicity of one kind of complexity over the other, for a genuine hardware implementation any of them poses an equally serious obstacle.

For the classes of FF-networks and benchmark tests considered, the networks with multiple hidden layers have no decisive superiority over the simple structures with fixed hidden units unless dimensionality of the input space is significant.

## 5 Appendix: Capacity and Function-Counting Theorem

The above results can be viewed as a step towards estimation of capacity of networks to memorise dichotomies. We intend to elaborate this subject further now and outline some of our recent results on this matter. A more detailed presentation will be available in future publications.

The capacity of a network in the sense of Cover [2] (*Cover's capacity*) is defined as a maximal $N$ such that for a randomly selected subset $\mathbf{X} \subset \mathbf{R}^n$ of $N$ points with probability 1 the network can implement $1/2$ of all dichotomies of $\mathbf{X}$. For a linear perceptron

$$F_{\mathbf{w}}(\mathbf{x}) \stackrel{\text{def}}{=} \sum_{i=1}^{k} w_i x_i \quad (\mathbf{x} \in \mathbf{X}), \tag{9}$$

where $\mathbf{w} \in \mathbf{R}^n$ is the vector of adjustable synaptic weights, the capacity is $2n$, and $2k$ for a FHU-network (1) with suitable chosen hidden units $\phi_1, ..., \phi_k$. These results are based on the so-called Function-Counting Theorem proved for the linear perceptron in the sixties (c.f. [2]). Extension of this result to the multilayer case is still an open problem (c.f. T. Cover's talk on NIPS'92). However, we have recently obtained the following partial result in this direction.

**Theorem 6** *Given a continuous probability density on $\mathbf{R}^n$, for a randomly selected subset $\mathbf{X} \subset \mathbf{R}^n$ of $N$ points the FF-network having the first hidden layer built from $h$ ThL-neurons can implement*

$$C(N, nh) \stackrel{\text{def}}{=} 2 \sum_{i=0}^{nh} \binom{N-1}{i}, \tag{10}$$

*dichotomies of $\mathbf{X}$ with a non-zero probability. Such a network can be constructed using $nh$ variable synaptic weights between input and hidden layer only.*

For $h = 1$ this theorem reduces to its classical form for which the phrase "with non-zero probability" can be strengthened to "with probability 1" [2].

The proof of the theorem develops Sakurai's idea of utilising the Vandermonde determinant to show the following property of the curve $c(t) \stackrel{\text{def}}{=} (t, t^2, ..., t^{n-1})$, $t > 0$

> (*) *for any subset $\mathbf{X}$ of $N$ points $\mathbf{x}_1 = c(t_1), ..., \mathbf{x}_N = c(t_N)$, $t_1 < t_2 < \cdots < t_N$, any hyperplane in $\mathbf{R}^n$ can intersect no more then $n$ different segments $[\mathbf{x}_i, \mathbf{x}_{i+1}]$ of $c$.*

The first step of the proof is to observe that the property (*) itself implies that the count (10) holds for such a set $X$. The second and the crucial step consists in showing that for a sufficiently small $\epsilon > 0$, for any selection of points $\tilde{\mathbf{x}}_1, ..., \tilde{\mathbf{x}}_N \in \mathbf{R}^n$ such that $||\tilde{\mathbf{x}}_i - \mathbf{x}_i|| < \epsilon$ for $i = 1, ..., n$, there exists a curve $\tilde{c}$ passing through these points and satisfying also the property (*).

Theorem 6 implies that in the class of multilayer FF-networks having the first hidden layer built from ThL-neurons only the single hidden layer networks are the most

efficient, since the higher layers have no influence on the number of implemented dichotomies (at least for the class of domains $x \subset \mathbf{R}^n$ considered).

Note that by virtue of (10) and the classical argument of Cover [2] for the class of domains $\mathbf{X}$ as in the Theorem 6 the capacity of the network considered is $2nh$. Thus the following estimates hold.

**Corollary 7** *In the class of FF-networks with a fixed number h of hidden units the ratio of the maximal capacity per hidden unit achievable by FHU-network to the maximal capacity per hidden unit achievable by VHU-networks having the ThL-neurons in the first hidden layer only is $2h/2nh = 1/n$. The analogous ratio for capacities per variable synaptic weight (in the class of FF-networks with a fixed number s of variable synaptic weights) is $\leq 2s/2s = 1$.*

**Acknowledgement.**    I thank A. Sakurai of Hitachi Ltd., for helpful comments leading to the improvement of results of the paper. The permission of the Director, Telecom Australia Research Laboratories, to publish this material is gratefully acknowledged.

# References

[1] E. Baum. On the capabilities of multilayer perceptrons. *Journal of Complexity*, 4:193–215, 1988.

[2] T.M. Cover. Geometrical and statistical properties of linear inequalities with applications to pattern recognition. *IEEE Trans. Elec. Comp.*, EC-14:326–334, 1965.

[3] R.M. Dudley. Central limit theorems for empirical measures. *Ann. Probability*, 6:899–929, 1978.

[4] F. Girosi and T. Poggio. Representation properties of networks: Kolmogorov's theorem is irrelevant. *Neural Computation*, 1:465–469, (1989).

[5] M. Golubitsky and V. Guillemin. *Stable Mapping and Their Singularities*. Springer-Verlag, New York, 1973.

[6] S. Huang and Y. Huang. Bounds on the number of hidden neurons in multilayer perceptrons. *IEEE Transactions on Neural Networks*, 2:47–55, (1991).

[7] V. Kurkova. Kolmogorov theorem is relevant. *Neural Computation*, 1, 1992.

[8] D. Psaltis, C.H. Park, and J. Hong. Higher order associative memories and their optical implementations. *Neural Networks*, 1:149–163, (1988).

[9] N. Redding, A. Kowalczyk, and T. Downs. Higher order separability and minimal hidden-unit fan-in. In T. Kohonen *et al.*, editor, *Artificial Neural Networks*, volume 1, pages 25–30. Elsevier, 1991.

[10] A. Sakurai. n-h-1 networks store no less $n \cdot h + 1$ examples but sometimes no more. In *Proceedings of IJCNN92*, pages III–936–III–941. IEEE, June 1992.
